# Effective Training of a Neural Network Character Classifier for Word Recognition

**Larry Yaeger**
Apple Computer
5540 Bittersweet Rd.
Morgantown, IN 46160
larryy@apple.com

**Richard Lyon**
Apple Computer
1 Infinite Loop, MS301-3M
Cupertino, CA 95014
lyon@apple.com

**Brandyn Webb**
The Future
4578 Fieldgate Rd.
Oceanside, CA 92056
brandyn@brainstorm.com

### Abstract

We have combined an artificial neural network (ANN) character classifier with context-driven search over character segmentation, word segmentation, and word recognition hypotheses to provide robust recognition of hand-printed English text in new models of Apple Computer's Newton MessagePad. We present some innovations in the training and use of ANNs as character classifiers for word recognition, including normalized output error, frequency balancing, error emphasis, negative training, and stroke warping. A recurring theme of reducing *a priori* biases emerges and is discussed.

## 1 INTRODUCTION

We have been conducting research on bottom-up classification techniques based on trainable artificial neural networks (ANNs), in combination with comprehensive but weakly-applied language models. To focus our work on a subproblem that is tractable enough to lead to usable products in a reasonable time, we have restricted the domain to hand-printing, so that strokes are clearly delineated by pen lifts. In the process of optimizing overall performance of the recognizer, we have discovered some useful techniques for architecting and training ANNs that must participate in a larger recognition process. Some of these techniques—especially the normalization of output error, frequency balancing, and error emphasis—suggest a common theme of significant value derived by reducing the effect of *a priori* biases in training data to better represent low frequency, low probability samples, including second and third choice probabilities.

There is ample prior work in combining low-level classifiers with various search strategies to provide integrated segmentation and recognition for writing (Tappert *et al* 1990) and speech (Renals *et al* 1992). And there is a rich background in the use of ANNs as classifiers, including their use as a low-level, character classifier in a higher-level word recognition system (Bengio *et al* 1995). But many questions remain regarding optimal strategies for deploying and combining these methods to achieve acceptable (to a real user) levels of performance. In this paper, we survey some of our experiences in exploring refinements and improvements to these techniques.

## 2 SYSTEM OVERVIEW

Our recognition system, the Apple-Newton Print Recognizer (ANPR), consists of three conceptual stages: Tentative Segmentation, Classification, and Context-Driven Search. The primary data upon which we operate are simple sequences of (*x,y*) coordinate pairs,

plus pen-up/down information, thus defining stroke primitives. The Segmentation stage decides which strokes will be combined to produce *segments*—the tentative groupings of strokes that will be treated as possible characters—and produces a sequence of these segments together with legal transitions between them. This process builds an implicit graph which is then scored in the Classification stage and examined for a maximum likelihood interpretation in the Search stage.

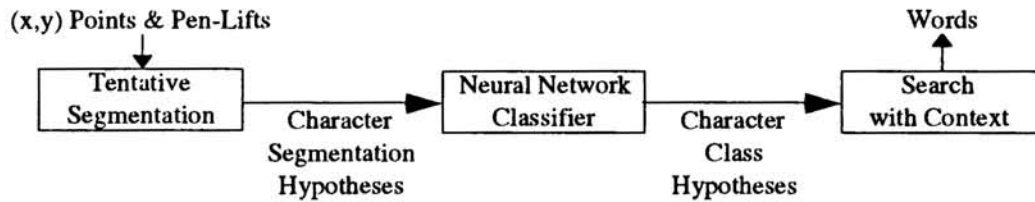

Figure 1: A Simplified Block Diagram of Our Hand-Print Recognizer.

# 3 TRAINING THE NEURAL NETWORK CLASSIFIER

Except for an *integrated multiple-representations* architecture (Yaeger *et al* 1996) and the training specifics detailed here, a fairly standard multi-layer perceptron trained with BP provides the ANN character classifier at the heart of ANPR. Training an ANN character classifier for use in a word recognition system, however, has different constraints than would training such a system for stand-alone character recognition. All of the techniques below, except for the annealing schedule, at least modestly *reduce* individual character recognition accuracy, yet dramatically *increase* word recognition accuracy.

A large body of prior work exists to indicate the general applicability of ANN technology as classifiers providing good estimates of *a posteriori* probabilities of each class given the input (Gish 1990, Richard and Lippmann 1991, Renals and Morgan 1992, Lippmann 1994, Morgan and Bourlard 1995, and others cited herein).

## 3.1 NORMALIZING OUTPUT ERROR

Despite their ability to provide good first choice *a posteriori* probabilities, we have found that ANN classifiers do a poor job of representing second and third choice probabilities when trained in the classic way—minimizing mean squared error for target vectors that are all 0's, except for a single 1 corresponding to the target class. This results in erratic word recognition failures as the net fails to accurately represent the legitimate ambiguity between characters. We speculated that reducing the "pressure towards 0" relative to the "pressure towards 1" as seen at the output units, and thus reducing the large bias towards 0 in target vectors, might permit the net to better model these inherent ambiguities.

We implemented a technique for "normalizing output error" (*NormOutErr*) by reducing the BP error for non-target classes relative to the target class by a factor that normalizes the total non-target error seen at a given output unit relative to the total target error seen at that unit. Assuming a training set with equal representation of classes, this normalization should then be based on the number of non-target versus target classes in a typical training vector, or, simply, the number of output units (minus one). Hence for *non-target* output units, we scale the error at each unit by a constant:

$$e' = Ae$$

where *e* is the error at an output unit, and *A* is defined to be:

$$A = 1 / \left[ d(N_{outputs} - 1) \right]$$

where $N_{outputs}$ is the number of output units, and $d$ is a user-adjusted tuning parameter, typically ranging from 0.1 to 0.2. Error at the *target* output unit is unchanged. Overall, this raises the activation values at the output units, due to the reduced pressure towards zero, particularly for low-probability samples. Thus the learning algorithm no longer

converges to a minimum mean-squared error (MMSE) estimate of $P(class|input)$, but to an MMSE estimate of a nonlinear function $f(P(class|input), A)$ depending on the factor $A$ by which we reduced the error pressure toward zero.

Using a simple version of the technique of Bourlard and Wellekens (1990), we worked out what that resulting nonlinear function is. The net will attempt to converge to minimize the modified quadratic error function

$$\left\langle \hat{E}^2 \right\rangle = p(1-y)^2 + A(1-p)y^2$$

by setting its output $y$ for a particular class to

$$y = p/(A - Ap + p)$$

where $p = P(class|input)$, and $A$ is as defined above. The inverse function is

$$p = yA/(yA + 1 - y)$$

We verified the fit of this function by looking at histograms of character-level empirical percentage-correct versus $y$, as in Figure 2.

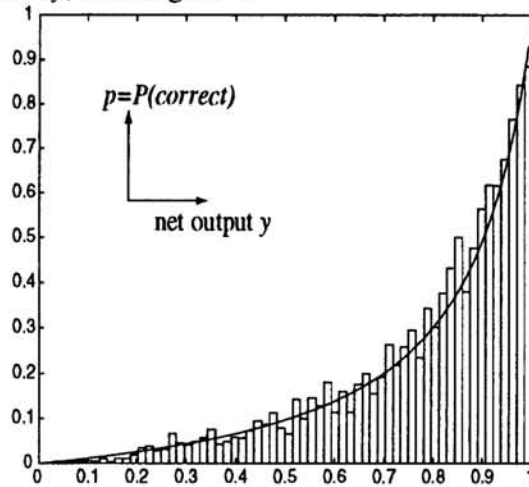

Figure 2: Empirical $p$ vs. $y$ Histogram for a Net Trained with $A$=0.11 ($d$=.1), with the Corresponding Theoretical Curve.

Note that the lower-probability samples have their output activations raised significantly, relative to the 45° line that $A = 1$ yields.

The primary benefit derived from this technique is that the net does a much better job of representing second and third choice probabilities, and low probabilities in general. Despite a small drop in top choice character accuracy when using NormOutErr, we obtain a very significant increase in word accuracy by this technique. Figure 3 shows an exaggerated example of this effect, for an atypically large value of $d$ (0.8), which overly penalizes character accuracy; however, the 30% decrease in word error rate is normal for this technique. (Note: These data are from a multi-year-old experiment, and are not necessarily representative of current levels of performance on any absolute scale.)

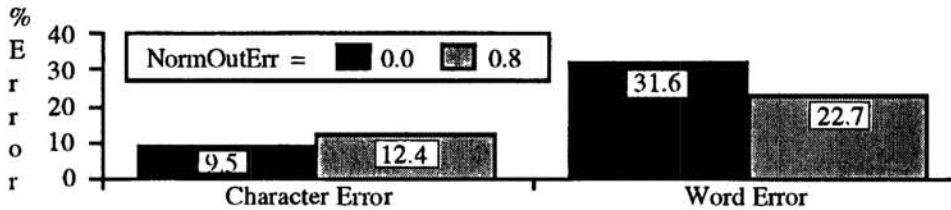

Figure 3: Character and Word Error Rates for Two Different Values of NormOutErr ($d$). A Value of 0.0 Disables NormOutErr, Yielding Normal BP. The Unusually High Value of 0.8 ($A$=0.013) Produces Nearly Equal Pressures Towards 0 and 1.

## 3.2 FREQUENCY BALANCING

Training data from natural English words and phrases exhibit very non-uniform priors for the various character classes, and ANNs readily model these priors. However, as with NormOutErr, we find that reducing the effect of these priors on the net, in a controlled way, and thus forcing the net to allocate more of its resources to low-frequency, low-probability classes is of significant benefit to the overall word recognition process. To this end, we explicitly (partially) balance the frequencies of the classes during training. We do this by probabilistically skipping and repeating patterns, based on a precomputed *repetition factor*. Each presentation of a repeated pattern is "warped" uniquely, as discussed later.

To compute the repetition factor for a class $i$, we first compute a normalized frequency of that class:

$$F_i = S_i / \overline{S}$$

where $S_i$ is the number of samples in class $i$, and $\overline{S}$ is the average number of samples over all classes, computed in the obvious way:

$$\overline{S} = (\frac{1}{C}\sum_{i=1}^{C} S_i)$$

with $C$ being the number of classes. Our repetition factor is then defined to be:

$$R_i = (a/F_i)^b$$

with $a$ and $b$ being user controls over the amount of skipping vs. repeating and the degree of prior normalization, respectively. Typical values of $a$ range from 0.2 to 0.8, while $b$ ranges from 0.5 to 0.9. The factor $a < 1$ lets us do more skipping than repeating; e.g. for $a = 0.5$, classes with relative frequency equal to half the average will neither skip nor repeat; more frequent classes will skip, and less frequent classes will repeat. A value of 0.0 for $b$ would do nothing, giving $R_i = 1.0$ for all classes, while a value of 1.0 would provide "full" normalization. A value of $b$ somewhat less than one seems to be the best choice, letting the net keep some bias in favor of classes with higher prior probabilities.

This explicit prior-bias reduction is related to Lippmann's (1994) and Morgan and Bourlard's (1995) recommended method for converting from the net's estimate of posterior probability, *p(class|input)*, to the value needed in an HMM or Viterbi search, *p(input|class)*, which is to divide by *p(class)* priors. Besides eliminating potentially noisy estimates of low probability classes and a possible need for renormalization, our approach forces the net to actually learn a better model of these lower frequency classes.

## 3.3 ERROR EMPHASIS

While frequency balancing corrects for under-represented classes, it cannot account for under-represented writing styles. We utilize a conceptually related probabilistic skipping of patterns, but this time for just those patterns that the net correctly classifies in its forward/recognition pass, as a form of "error emphasis", to address this problem. We define a *correct-train probability* (0.1 to 1.0) that is used as a biased coin to determine whether a particular pattern, having been correctly classified, will also be used for the backward/training pass or not. This only applies to correctly segmented, or "positive" patterns, and misclassified patterns are never skipped.

Especially during early stages of training, we set this parameter fairly low (around 0.1), thus concentrating most of the training time and the net's learning capability on patterns that are more difficult to correctly classify. This is the only way we were able to get the net to learn to correctly classify unusual character variants, such as a 3-stroke "5" as written by only one training writer.

Variants of this scheme are possible in which misclassified patterns would be repeated, or different learning rates would apply to correctly and incorrectly classified patterns. It is also related to techniques that use a training subset, from which easily-classified patterns are replaced by randomly selected patterns from the full training set (Guyon *et al* 1992).

## 3.4 NEGATIVE TRAINING

Our recognizer's tentative segmentation stage necessarily produces a large number of invalid segments, due to inherent ambiguities in the character segmentation process. During recognition, the network must classify these invalid segments just as it would any valid segment, with no knowledge of which are valid or invalid. A significant increase in word-level recognition accuracy was obtained by performing *negative training* with these invalid segments. This consists of presenting invalid segments to the net during training, with all-zero target vectors. We retain control over the degree of negative training in two ways. First is a *negative-training factor* (0.2 to 0.5) that modulates the learning rate (equivalently by modulating the error at the output layer) for these negative patterns. This reduces the impact of negative training on positive training, and modulates the impact on characters that specifically look like elements of multi-stroke characters (e.g., I, 1, l, o, O, 0). Secondly, we control a *negative-training probability* (0.05 to 0.3), which determines the probability that a particular negative sample will actually be trained on (for a given presentation). This both reduces the overall impact of negative training, and significantly reduces training time, since invalid segments are more numerous than valid segments. As with NormOutErr, this modification hurts character-level accuracy a little bit, but helps word-level accuracy a lot.

## 3.5 STROKE WARPING

During training (but not during recognition), we produce random variations in stroke data, consisting of small changes in skew, rotation, and $x$ and $y$ linear and quadratic scalings. This produces alternate character forms that are consistent with stylistic variations within and between writers, and induces an explicit aspect ratio and rotation invariance within the framework of standard back-propagation. The amounts of each distortion to apply were chosen through cross-validation experiments, as just the amount needed to yield optimum generalization. We also examined a number of such samples by eye to verify that they represent a natural range of variation. A small set of such variations is shown in Figure 4.

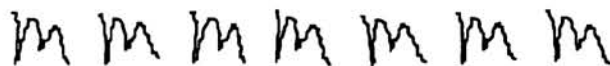

Figure 4:  A Few Random Stroke Warpings of the Same Original "m" Data.

Our stroke warping scheme is somewhat related to the ideas of Tangent Dist and Tangent Prop (Simard *et al* 1992, 1993), in terms of the use of predetermined families of transformations, but we believe it is much easier to implement. It is also somewhat distinct in applying transformations on the original coordinate data, as opposed to using distortions of images. The voice transformation scheme of Chang and Lippmann (1995) is also related, but they use a static replication of the training set through a small number of transformations, rather than dynamic random transformations of infinite variety.

## 3.6 ANNEALING & SCHEDULING

Many discussions of back-propagation seem to assume the use of a single fixed learning rate. We view the stochastic back-propagation process as more of a simulated annealing, with a learning rate starting very high and decreasing only slowly to a very low value. We typically start with a rate near 1.0 and reduce the rate by a *decay factor* of 0.9 until it gets down to about 0.001. The rate decay factor is applied following any epoch for which the total squared error increased on the training set. Repeated tests indicate that this

approach yields better results than low (or even moderate) initial learning rates, which we speculate to be related to a better ability to escape local minima.

In addition, we find that we obtain best overall results when we also allow some of our many training parameters to change over the course of a training run. In particular, the correct train probability needs to start out very low to give the net a chance to learn unusual character styles, but it should finish up at 1.0 in order to not introduce a general posterior probability bias in favor of classes with lots of ambiguous examples. We typically train a net in four "phases" according to parameters such as in Figure 5.

| Phase | Epochs | Learning Rate | Correct Train Prob | Negative Train Prob |
|-------|--------|---------------|--------------------|---------------------|
| 1 | 25 | 1.0 - 0.5 | 0.1 | 0.05 |
| 2 | 25 | 0.5 - 0.1 | 0.25 | 0.1 |
| 3 | 50 | 0.1 - 0.01 | 0.5 | 0.18 |
| 4 | 30 | 0.01 - 0.001 | 1.0 | 0.3 |

Figure 5: A Typical Multi-Phase Schedule of Learning Rates and Other Parameters for Training a Character-Classifier Net.

## 4 DISCUSSION AND FUTURE DIRECTIONS

The normalization of output error, frequency balancing, and error emphasis network-training methods discussed previously share a unifying theme: Reducing the effect of *a priori* biases in the training data on network learning significantly improves the network's performance in an integrated recognition system, despite a modest reduction in the network's accuracy for individual characters. Normalization of output error prevents over-represented non-target classes from biasing the net against under-represented target classes. Frequency balancing prevents over-represented target classes from biasing the net against under-represented target classes. And error emphasis prevents over-represented writing styles from biasing the net against under-represented writing styles. One could even argue that negative training eliminates an absolute bias towards properly segmented characters, and that stroke warping reduces the bias towards those writing styles found in the training data, although these techniques provide wholly new information to the system as well.

Though we've offered arguments for why each of these techniques, individually, helps the overall recognition process, it is unclear why prior-bias reduction, in general, should be so consistently valuable. The general effect may be related to the technique of dividing out priors, as is sometimes done to convert from *p(class|input)* to *p(input|class)*. But we also believe that forcing the net, during learning, to allocate resources to represent less frequent sample types may be directly beneficial. In any event, it is clear that paying attention to such biases and taking steps to modulate them is a vital component of effective training of a neural network serving as a classifier in a maximum likelihood recognition system.

We speculate that a method of modulating the learning rate at each output unit—based on a measure of its accuracy relative to the other output units—may be possible, and that such a method might yield the combined benefits of several of these techniques, with fewer user-controllable parameters.

## Acknowledgements

This work was done in collaboration with Bill Stafford, Apple Computer, and Les Vogel, Angel Island Technologies. We are also indebted to our many colleagues in the connectionist community and at Apple Computer.

Some techniques in this paper have pending U.S. and foreign patent applications.

## References

Y. Bengio, Y. LeCun, C. Nohl, and C. Burges, "LeRec: A NN/HMM Hybrid for On-Line Handwriting Recognition," Neural Computation, Vol. 7, pp. 1289-1303, 1995.

H. Bourlard and C. J. Wellekens, "Links between Markov Models and Multilayer Perceptrons," *IEEE Trans. PAMI*, Vol. 12, pp. 1167–1178, 1990.

E. I. Chang and R. P. Lippmann, "Using Voice Transformations to Create Additional Training Talkers for Word Spotting," in *Advances in Neural Information Processing Systems 7*, Tesauro et al. (eds.), pp. 875–882, MIT Press, 1995.

H. Gish, "A Probabilistic Approach to Understanding and Training of Neural Network Classifiers," *Proc. IEEE Intl. Conf. on Acoustics, Speech, and Signal Processing* (Albuquerque, NM), pp. 1361–1364, 1990.

I. Guyon, D. Henderson, P. Albrecht, Y. LeCun, and P. Denker, "Writer independent and writer adaptive neural network for on-line character recognition," in *From pixels to features III*, S. Impedovo (ed.), pp. 493–506, Elsevier, Amsterdam, 1992.

R. A. Jacobs, M. I. Jordan, S. J. Nowlan, and G. E. Hinton, "Adaptive Mixtures of Local Experts," *Neural Computation*, Vol. 3, pp. 79–87, 1991.

R. P. Lippmann, "Neural Networks, Bayesian *a posteriori* Probabilities, and Pattern Classification," pp. 83–104 in: *From Statistics to Neural Networks—Theory and Pattern Recognition Applications*, V. Cherkassky, J. H. Friedman, and H. Wechsler (eds.), Springer-Verlag, Berlin, 1994.

N. Morgan and H. Bourlard, "Continuous Speech Recognition—An introduction to the hybrid HMM/connectionist approach," IEEE Signal Processing Mag., Vol. 13, no. 3, pp. 24–42, May 1995.

S. Renals and N. Morgan, "Connectionist Probability Estimation in HMM Speech Recognition," TR-92-081, International Computer Science Institute, 1992.

S. Renals and N. Morgan, M. Cohen, and H. Franco "Connectionist Probability Estimation in the Decipher Speech Recognition System," *Proc. IEEE Intl. Conf. on Acoustics, Speech, and Signal Processing* (San Francisco), pp. I-601–I-604, 1992.

M. D. Richard and R. P. Lippmann, "Neural Network Classifiers Estimate Bayesian *a Posteriori* Probabilities," *Neural Computation*, Vol. 3, pp. 461–483, 1991.

P. Simard, B. Victorri, Y. LeCun and J. Denker, "Tangent Prop—A Formalism for Specifying Selected Invariances in an Adaptive Network," in *Advances in Neural Information Processing Systems 4*, Moody et al. (eds.), pp. 895–903, Morgan Kaufmann, 1992.

P. Simard, Y. LeCun and J. Denker, "Efficient Pattern Recognition Using a New Transformation Distance," in *Advances in Neural Information Processing Systems 5*, Hanson et al. (eds.), pp. 50–58, Morgan Kaufmann, 1993.

C. C. Tappert, C. Y. Suen, and T. Wakahara, "The State of the Art in On-Line Handwriting Recognition," *IEEE Trans. PAMI*, Vol. 12, pp. 787–808, 1990.

L. Yaeger, B. Webb, and R. Lyon, "Combining Neural Networks and Context-Driven Search for On-Line, Printed Handwriting Recognition", (unpublished).
